# Feature Correspondence:
# A Markov Chain Monte Carlo Approach

**Frank Dellaert, Steven M. Seitz, Sebastian Thrun, and Charles Thorpe**
Department of Computer Science &Robotics Institute
Carnegie Mellon University
Pittsburgh, PA 15213
{dellaert,seitz,thrun,cet}@cs.cmu.edu

## Abstract

When trying to recover 3D structure from a set of images, the most difficult problem is establishing the correspondence between the measurements. Most existing approaches assume that features can be tracked across frames, whereas methods that exploit rigidity constraints to facilitate matching do so only under restricted camera motion. In this paper we propose a Bayesian approach that avoids the brittleness associated with singling out one "best" correspondence, and instead consider the distribution over all possible correspondences. We treat both a fully Bayesian approach that yields a posterior distribution, and a MAP approach that makes use of EM to maximize this posterior. We show how Markov chain Monte Carlo methods can be used to implement these techniques in practice, and present experimental results on real data.

## 1  Introduction

Structure from motion (SFM) addresses the problem of simultaneously recovering camera pose and a three-dimensional model from a collection of images. This problem has received considerable attention in the computer vision community [1, 2, 3]. Methods that can robustly reconstruct the 3D structure of environments have a potentially large impact in many areas of societal importance, such as architecture, entertainment, space exploration and mobile robotics.

A fundamental problem in SFM is *data association*, i.e., the question of determining correspondence between features observed in different images. This problem has been referred to as the most difficult part of structure recovery [4], and is particularly challenging if the images have been taken from widely separated viewpoints. Virtually all existing approaches assume that either the correspondence is known *a priori*, or that features can be tracked from frame to frame [1, 2]. Methods based on the robust recovery of epipolar geometry [3, 4] can cope with larger inter-frame displacements, but still depend on the ability to identify a set of initial correspondences to seed the robust matching process. In this paper, we are interested in cases where individual camera images are recorded from vastly different viewpoints, which renders existing SFM approaches inapplicable. Traditional approaches for

establishing correspondence between sets of 2D features [5, 6, 7] are of limited use in this domain, as the projected 3D structure can look very different in each image.

This paper proposes a Bayesian approach to data association. Instead of considering a single correspondence only (which we conjecture to be brittle), our approach considers whole distributions over correspondences. As a result, our approach is more robust, and from a Bayesian perspective it is also sound. Unfortunately, no closed-form solution exists for calculating these distributions conditioned on the camera images. Therefore, we propose to use the Metropolis-Hastings algorithm, a popular Markov chain Monte Carlo (MCMC) method, to sample from the posterior.

In particular, we propose two different algorithms. The first method, discussed in Section 2, is mathematically more powerful but computationally expensive. It uses MCMC to sample from the joint distribution over both correspondences and three-dimensional scene structure. While this approach is mathematically elegant from a Bayesian point of view, we have so far only been able to obtain results for simple, artificial domains. Thus, to cope with large-scale data sets, we propose in Section 3 a *maximum a posteriori* (MAP) approach using the Expectation-Maximization (EM) algorithm to maximize the posterior. Here we use MCMC sampling only for the data association problem. Simulated annealing is used to reduce the danger of getting stuck in local minima. Experimental results obtained in realistic domains and presented in Section 4 suggest that this approach works well in the general SFM case, and that it scales favorably to complex computer vision problems.

The idea of using MCMC for data association has been used before by [8] in the context of a traffic surveillance application. However, their approach is not directly applicable to SFM, as the computer vision domain is characterized by a large number of local minima. Our paper goes beyond theirs in two important aspects: First, we develop a framework for MCMC sampling over both the data association and the model, and second, we apply annealing to smooth the posterior so as to reduce the chance to get stuck in local minima. In a previous paper [9] we have discussed the idea of using EM for SFM, but without the unifying framework presented below.

## 2 A Fully Bayesian Approach using MCMC

Below we derive the general approach for MCMC sampling from the joint posterior over data association and models. We only show results for a simple example from pose estimation, as this approach is computationally very demanding. An EM approach based on the general principles described here, but applicable to larger-scale problems, will be described in the next section.

### 2.1 Structure from Motion

The *structure from motion* problem is this: given a set of images of a scene, taken from different viewpoints, recover the 3D structure of the scene along with the camera parameters. In the feature-based approach to SFM, we consider the situation in which a set of $N$ 3D features $\mathbf{x}_j$ is viewed by a set of $m$ cameras with parameters $\mathbf{m}_i$. As input data we are given the set of 2D measurements $\mathbf{u}_{ik}$ in the images, where $k \in \{1..K_i\}$ and $K_i$ is the number of measurements in the $i$-th image. To model correspondence information, we introduce for each measurement $\mathbf{u}_{ik}$ the indicator variable $\mathbf{j}_{ik}$, indicating that $\mathbf{u}_{ik}$ is a measurement of the $\mathbf{j}_{ik}$-th feature $\mathbf{x}_{\mathbf{j}_{ik}}$.

The choice of feature type and camera model determines the *measurement function* $\mathbf{h}(\mathbf{m}_i, \mathbf{x}_j)$, predicting the measurement $\mathbf{u}_{ik}$ given $\mathbf{m}_i$ and $\mathbf{x}_j$ (with $j = \mathbf{j}_{ik}$):

$$\mathbf{u}_{ik} = \mathbf{h}(\mathbf{m}_i, \mathbf{x}_j) + \mathbf{n}$$

where $\mathbf{n}$ is the measurement noise. Without loss of generality, let us consider the case in which the features $\mathbf{x}_j$ are 3D points and the measurements $\mathbf{u}_{ik}$ are points in the 2D image. In this case the measurement function can be written as a 3D rigid displacement followed by a projection:

$$\mathbf{h}(\mathbf{m}_i, \mathbf{x}_j) = \Phi[\mathbf{R}_i(\mathbf{x}_j - \mathbf{t}_i)] \tag{1}$$

where $\mathbf{R}_i$ and $\mathbf{t}_i$ are the rotation matrix and translation of the $i$-th camera, respectively, and $\Phi : \mathbb{R}^3 \to \mathbb{R}^2$ is the camera projection model.

## 2.2 Deriving the Posterior

Whereas previous methods single out a single "best" correspondence across images, in a Bayesian framework we are interested in characterizing our knowledge about the unknowns conditioned on the data *only*, averaging over all possible correspondences. Thus, we are interested in the posterior distribution $P(\theta|\mathbf{U})$, where $\theta$ collects the unknown model parameters $\mathbf{m}_i$ and $\mathbf{x}_j$. In the case of unknown correspondence, we need to sum over all possible assignments $\mathbf{J} = \{\mathbf{j}_{ik}\}$ to obtain

$$P(\theta|\mathbf{U}) = \sum_{\mathbf{J}} P(\mathbf{J}, \theta|\mathbf{U}) \propto P(\theta) \sum_{\mathbf{J}} P(\mathbf{U}|\mathbf{J}, \theta) P(\mathbf{J}|\theta) \tag{2}$$

where we have applied Bayes law and the chain rule. Let us assume for now that there are no occlusions or spurious measurements, so that $K_i = N$ and $\mathbf{J}$ is a set of $m$ permutations $\mathbf{J}_i$ of the indices $1..N$. Then, assuming i.i.d. normally distributed noise on the measurements, each term in (2) can be calculated using

$$P(\mathbf{J}|\theta) = (\tfrac{1}{N!})^m \qquad P(\mathbf{U}|\mathbf{J}, \theta) = \prod_{i=1}^m \prod_{k=1}^{K_i} \mathcal{N}(\mathbf{u}_{ik}; \mathbf{h}(\mathbf{m}_i, \mathbf{x}_{\mathbf{j}_{ik}}), \sigma) \tag{3}$$

if each $\mathbf{J}_i$ is a permutation, and 0 otherwise. Here $\mathcal{N}(.; \mu, \sigma)$ denotes the normal distribution with mean $\mu$ and standard deviation $\sigma$. The first identity in (3) holds if we assume each of the $N!$ possible permutations to be equally likely *a priori*.

## 2.3 Sampling from the Posterior using MCMC

Unfortunately, direct computation of the total posterior distribution $P(\theta|\mathbf{U})$ in (2) is intractable in general, because the number of correspondence assignments $\mathbf{J}$ is combinatorial in the number of features and images. As a solution to this computational challenge we propose to instead *sample* from $P(\theta|\mathbf{U})$. Sampling directly from $P(\theta|\mathbf{U})$ is equally difficult, but if we can obtain a sample $\{(\theta^{(r)}, \mathbf{J}^{(r)})\}$ from the *joint* distribution $P(\theta, \mathbf{J}|\mathbf{U})$, we can simply discard the correspondence part $\mathbf{J}^{(r)}$ to obtain a sample $\{\theta^{(r)}\}$ from the marginal distribution $P(\theta|\mathbf{U})$.

To sample from the joint distribution $P(\theta, \mathbf{J}|\mathbf{U})$ we propose to use MCMC sampling, in particular the *Metropolis-Hastings* algorithm [10]. This method involves simulating a Markov chain whose equilibrium distribution is the desired posterior distribution $P(\theta, \mathbf{J}|\mathbf{U})$. Defining $\mathbf{X} \triangleq (\mathbf{J}, \theta)$, the algorithm is:

1. Start with a random initial state $\mathbf{X}^{(0)}$.
2. Propose a new state $\mathbf{X}'$ using a chosen *proposal density* $Q(\mathbf{X}'; \mathbf{X}^{(r)})$.
3. Compute the ratio

$$a = \frac{P(\mathbf{X}'|\mathbf{U})}{P(\mathbf{X}^{(r)}|\mathbf{U})} \frac{Q(\mathbf{X}^{(r)}; \mathbf{X}')}{Q(\mathbf{X}'; \mathbf{X}^{(r)})} \tag{4}$$

4. Accept $\mathbf{X}'$ as $\mathbf{X}^{(r+1)}$ with probability $\min(a, 1)$, otherwise $\mathbf{X}^{(r+1)} = \mathbf{X}^{(r)}$.

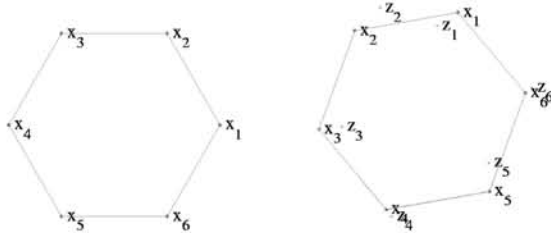

Figure 1: Left: A 2D model shape, defined by the 6 feature points $\mathbf{x}_j$. Right: Transformed shape (by a simple rotation) and 6 noisy measurements $\mathbf{u}_k$ of the transformed features. The true rotation is 70 degrees, noise is zero-mean Gaussian.

The sequence of tuples $(\theta^{(r)}, \mathbf{J}^{(r)})$ thus generated will be a sample from $P(\theta, \mathbf{J}|\mathbf{U})$, if the sampler is run sufficiently long. To calculate the acceptance ratio $a$, we assume that the noise on the feature measurements is normally distributed and isotropic. Using Bayes law and eq. (3), we can then rewrite $a$ from (4) as

$$a = \frac{\prod_{i=1}^{m} \prod_{k=1}^{K_i} \mathcal{N}(\mathbf{u}_{ik}; \mathbf{h}(\mathbf{m}_i', \mathbf{x}_{\mathbf{j}_{ik}}'), \sigma)}{\prod_{i=1}^{m} \prod_{k=1}^{K_i} \mathcal{N}(\mathbf{u}_{ik}; \mathbf{h}(\mathbf{m}_i^{(r)}, \mathbf{x}_{\mathbf{j}_{ik}}^{(r)}), \sigma)} \frac{Q(\mathbf{X}^{(r)}; \mathbf{X}')}{Q(\mathbf{X}'; \mathbf{X}^{(r)})}$$

Simplifying the notation by defining $\mathbf{h}_{ik}^{(r)} \triangleq \mathbf{h}(\mathbf{m}_i^{(r)}, \mathbf{x}_{\mathbf{j}_{ik}}^{(r)})$, we obtain

$$a = \frac{Q(\mathbf{X}^{(r)}; \mathbf{X}')}{Q(\mathbf{X}'; \mathbf{X}^{(r)})} \exp\left[ \frac{1}{2\sigma^2} \sum_{i,k} (\|\mathbf{u}_{ik} - \mathbf{h}_{ik}^{(r)}\|^2 - \|\mathbf{u}_{ik} - \mathbf{h}_{ik}'\|^2) \right] \qquad (5)$$

The proposal density $Q(.;.)$ is application dependent, and an example is given below.

## 2.4  Example: A 2D Pose Estimation Problem

To illustrate this method, we present a simple example from pose estimation. Assume we have a 2D model shape, given in the form of a set of 2D points $\mathbf{x}_j$, as shown in Figure 1. We observe an image of this shape which has undergone a rotation $\theta$ to be estimated. This rotated shape is shown at right in the figure, along with 6 noisy measurements $\mathbf{u}_k$ on the feature points. In Figure 2 at left we show the posterior distribution over the rotation parameter, given the measurements from Figure 1 and with *known* correspondence. In this case, the posterior is unimodal. In the case of unknown correspondence, the posterior *conditioned on the data alone* is shown at right in Figure 2 and is a mixture of 6!=720 functions of the form (3), with 6 equally likely modes induced by the symmetry of the model shape.

In order to perform MCMC sampling, we implement the proposal step by choosing randomly between two strategies. (a) In a "small perturbation" we keep the correspondence assignment $\mathbf{J}$ but add a small amount of noise to $\theta$. This serves to explore the values of $\theta$ within a mode of the posterior probability. (b) In a "long jump", we completely randomize both $\theta$ and $\mathbf{J}$. This provides a way to jump between probability modes. Note that $Q(\mathbf{X}^{(r)}; \mathbf{X}')/Q(\mathbf{X}'; \mathbf{X}^{(r)}) = 1$ for this proposal density. The result of the sampling procedure is shown as a histogram of the rotation parameter $\theta$ in Figure 3. The histogram is a non-parametric approximation to the analytic posterior shown in Figure 2. The figure shows the results of running a sampler for 100,000 steps, the first 1000 of which were discarded as a transient. Note that even for this simple example, there is still considerable correlation in the sample

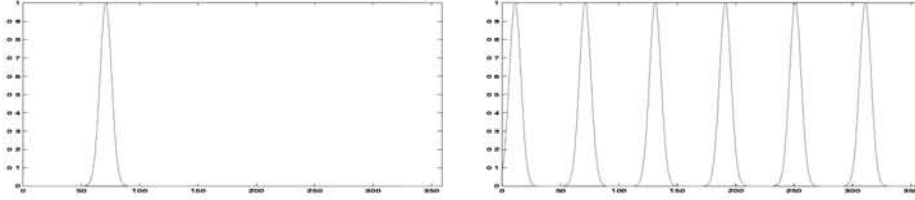

Figure 2: (Left) The posterior distribution over rotation $\theta$ with known correspondence, and (Right) with unknown correspondence, a mixture with 720 components.

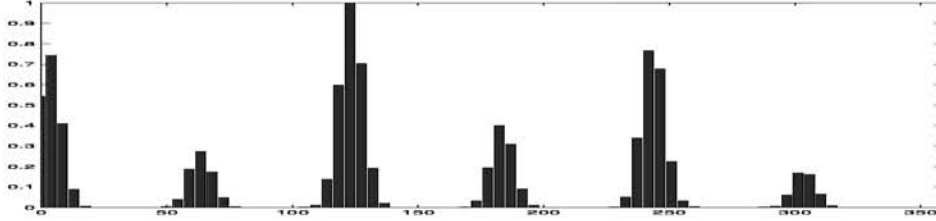

Figure 3: Histogram for the values of $\theta$ obtained in one MCMC run, for the situation in Figure 1. The MCMC sampler was run for 100,000 steps.

of 100,000 states as evidenced by the uneven mass in each of the 6 analytically predicted modes.

## 3    Maximum a Posteriori Estimation using MCEM

As illustrated above, sampling from the joint probability over assignments $\mathbf{J}$ and parameters $\theta$ using MCMC can be very expensive. However, if only a *maximum a posteriori* (MAP) estimate is needed, sampling over the joint space can be avoided by means of the EM algorithm. To obtain the MAP estimate, we need to maximize $P(\theta|\mathbf{U})$ as given by (2). This is intractable in general because of the combinatorial number of terms. The EM algorithm provides a tractable alternative to maximizing $P(\theta|\mathbf{U})$, using the correspondence $\mathbf{J}$ as a hidden variable [11]. It iterates over:

**E-step:** Calculate the expected log-posterior $Q^t(\boldsymbol{\Theta})$:

$$Q^t(\boldsymbol{\Theta}) \triangleq E_{\theta^t}\{\log P(\theta|\mathbf{U},\mathbf{J})|\mathbf{U}\} = \sum_{\mathbf{J}} P(\mathbf{J}|\mathbf{U},\theta^t) \log P(\theta|\mathbf{U},\mathbf{J}) \qquad (6)$$

where the expectation is taken with respect to the posterior distribution $P(\mathbf{J}|\mathbf{U},\theta^t)$ over all possible correspondence assignments $\mathbf{J}$ given the measurement data $\mathbf{U}$ and a *current guess* $\theta^t$ for the parameters.

**M-step:** Re-estimate $\theta^{t+1}$ by maximizing $Q^t(\boldsymbol{\Theta})$, i.e., $\theta^{t+1} = \text{argmax }_\theta Q^t(\boldsymbol{\Theta})$

Instead of calculating $Q^t(\boldsymbol{\Theta})$ exactly using (6), which again involves summing over a combinatorial number of terms, we can replace it by a Monte Carlo approximation:

$$Q^t(\boldsymbol{\Theta}) \approx \frac{1}{R} \sum_{r=1}^{R} \log P(\theta|\mathbf{U},\mathbf{J}^{(r)}) \qquad (7)$$

where $\{\mathbf{J}^{(r)}\}$ is a sample from $P(\mathbf{J}|\mathbf{U},\theta^t)$ obtained by MCMC sampling. Formally this can be justified in the context of a *Monte Carlo EM* or MCEM, a version

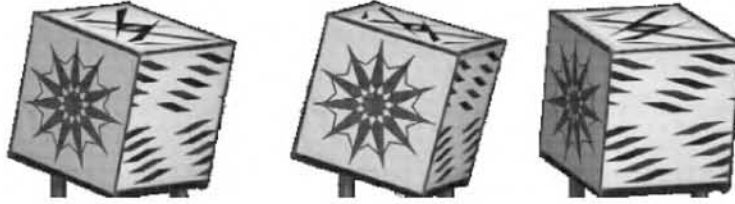

Figure 4: Three out of 11 *cube* images. Although the images were originally taken as a sequence in time, the ordering of the images is irrelevant to our method.

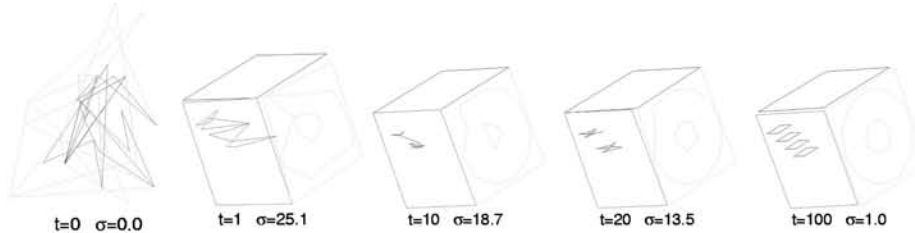

t=0  σ=0.0    t=1  σ=25.1    t=10  σ=18.7    t=20  σ=13.5    t=100  σ=1.0

Figure 5: Starting from random structure (t=0) we recover gross 3D structure in the very first iteration (t=1). As the annealing parameter $\sigma$ is gradually decreased, successively finer details are resolved (iterations 1,10,20, and 100 are shown).

of the EM algorithm where the E-step is executed by a Monte-Carlo process [11]. The sampling proceeds as in the previous section, using the Metropolis-Hastings algorithm, but now with a fixed parameter $\theta = \theta^t$. Note that at each iteration the estimate $\theta^t$ changes and we sample from a different posterior distribution $P(\mathbf{J}|\mathbf{U}, \theta^t)$.

In practice it is important to add *annealing* to this basic EM scheme, to avoid getting stuck in local minima. In simulated annealing we artificially increase the noise parameter $\sigma$ for the early iterations, gradually decreasing it to its correct value. This has two beneficial consequences. First, the posterior distribution $P(\mathbf{J}|\mathbf{U}, \theta^t)$ is less peaked when $\sigma$ is high, allowing the MCMC sampler to explore the space of assignments $\mathbf{J}$ more easily. Second, the expected log-posterior $Q^t(\Theta)$ is smoother and has fewer local maxima for higher values of $\sigma$.

## 4   Results

To validate our approach we have conducted a number of experiments, one of which is presented here. The input data in this experiment consisted of 55 manually selected measurements in each of 11 input images, three of which are shown in Figure 4. Note that features are not tracked from frame to frame and the images can be presented in arbitrary order. To initialize the 11 cameras $\mathbf{m}_i$ are all placed at the origin, looking towards the 55 model points $\mathbf{x}_j$, who themselves are normally distributed at unit distance from the cameras. We used an orthographic projection model. The EM algorithm was run for 100 iterations, and the sampler for 10000 steps per image. For this data set the algorithm took about a minute to complete on a standard PC.

The algorithm converges consistently and fast to an estimate for the structure and motion where the correct correspondence is the most probable one, and where all

assignments in the different images agree with each other. A typical run of the algorithm is shown in Figure 5, where we have shown a wireframe model of the recovered structure at several points during the run. There are two important points to note: (a) *the gross structure is recovered in the very first iteration, starting from random initial structure*, and (b) finer details of the structure are gradually resolved as the annealing parameter $\sigma$ is decreased. The estimate for the structure after convergence is almost identical to the one found by the factorization method [1] when this is provided with the correct correspondence.

## 5    Conclusions and Future Directions

In this paper we presented a theoretically sound method to deal with ambiguous feature correspondence, and have shown how Markov chain Monte Carlo sampling can be used to obtain practical algorithms. We have detailed this for two cases: (1) obtaining a posterior distribution over the parameters $\theta$, and (2) obtaining a MAP estimate by means of EM. In future work, we would like to apply these methods in other domains where data association plays a central role. In particular, in the highly active area of mobile robot mapping, the data association problem is currently a major obstacle to building large-scale maps [12, 13]. We conjecture that our approach is equally applicable to the robotic mapping problem, and can lead to qualitatively new solutions in that domain.

## References

[1] C. Tomasi and T. Kanade. Shape and motion from image streams under orthography: a factorization method. *Int. J. of Computer Vision*, 9(2):137–154, Nov. 1992.

[2] R.I. Hartley. Euclidean reconstruction from uncalibrated views. In *Application of Invariance in Computer Vision*, pages 237–256, 1994.

[3] P.A. Beardsley, P.H.S. Torr, and A. Zisserman. 3D model acquisition from extended image sequences. In *Eur. Conf. on Computer Vision (ECCV)*, pages II:683–695, 1996.

[4] P. Torr, A. Fitzgibbon, and A. Zisserman. Maintaining multiple motion model hypotheses over many views to recover matching and structure. In *Int. Conf. on Computer Vision (ICCV)*, pages 485–491, 1998.

[5] G.L. Scott and H.C. Longuet-Higgins. An algorithm for associating the features of two images. *Proceedings of Royal Society of London*, B-244:21–26, 1991.

[6] L.S. Shapiro and J.M. Brady. Feature-based correspondence: An eigenvector approach. *Image and Vision Computing*, 10(5):283–288, June 1992.

[7] S. Gold, A. Rangarajan, C. Lu, S. Pappu, and E. Mjolsness. New algorithms for 2D and 3D point matching. *Pattern Recognition*, 31(8):1019–1031, 1998.

[8] H. Pasula, S. Russell, M. Ostland, and Y. Ritov. Tracking many objects with many sensors. In *Int. Joint Conf. on Artificial Intelligence (IJCAI)*, Stockholm, 1999.

[9] F. Dellaert, S.M. Seitz, C.E. Thorpe, and S. Thrun. Structure from motion without correspondence. In *IEEE Conf. on Computer Vision and Pattern Recognition (CVPR)*, June 2000.

[10] W.R. Gilks, S. Richardson, and D.J. Spiegelhalter, editors. *Markov chain Monte Carlo in practice*. Chapman and Hall, 1996.

[11] M.A. Tanner. *Tools for Statistical Inference*. Springer, 1996.

[12] J.J. Leonard and H.J.S. Feder. A computationally efficient method for large-scale concurrent mapping and localization. In *Proceedings of the Ninth International Symposium on Robotics Research*, Salt Lake City, Utah, 1999.

[13] J.A. Castellanos and J.D. Tardós. *Mobile Robot Localization and Map Building: A Multisensor Fusion Approach*. Kluwer Academic Publishers, Boston, MA, 2000.
